# On the Separation of Signals from Neighboring Cells in Tetrode Recordings

**Maneesh Sahani, John S. Pezaris and Richard A. Andersen**
maneesh@caltech.edu, pz@caltech.edu, andersen@vis.caltech.edu
Computation and Neural Systems
California Institute of Technology
216-76 Caltech, Pasadena, CA 91125 USA

## Abstract

We discuss a solution to the problem of separating waveforms produced by multiple cells in an extracellular neural recording. We take an explicitly probabilistic approach, using latent-variable models of varying sophistication to describe the distribution of waveforms produced by a single cell. The models range from a single Gaussian distribution of waveforms for each cell to a mixture of hidden Markov models. We stress the overall statistical structure of the approach, allowing the details of the generative model chosen to depend on the specific neural preparation.

## 1 INTRODUCTION

Much of our empirical understanding of the systems-level functioning of the brain has come from a procedure called extracellular recording. The electrophysiologist inserts an insulated electrode with exposed tip into the extracellular space near one or more neuron cell bodies. Transient currents due to action potentials across nearby cell membranes are then recorded as deflections in potential, *spikes*, at the electrode tip. At an arbitrary location in gray matter, an extracellular probe is likely to see pertubations due to firing in many nearby cells, each cell exhibiting a distinct waveform due to the differences in current path between the cells and the electrode tip. Commonly, the electrode is maneuvered until all the recorded deflections have almost the same shape; the spikes are then all presumed to have arisen from a single *isolated* cell. This process of cell isolation is time-consuming, and it permits recording from only one cell at a time. If differences in spike waveform can be exploited to *sort* recorded events by cell, the experimental cost of extracellular recording can be reduced, and data on interactions between simultaneously recorded cells can be obtained.

Many *ad hoc* solutions to spike sorting have been proposed and implemented, but thus far an explicit statistical foundation, with its accompanying benefits, has mostly been lacking. Lewicki (1994) is the exception to this rule and provides a well-founded probabilistic approach, but uses assumptions (such as isotropic Gaussian variability) that are not well supported in many data sets (see Fee *et al* (1996)).

A first step in the construction of a solution to the spike-sorting problem is the specification of a model by which the data are taken to be generated. The model has to be powerful enough to account for most of the variability observed in the data, while being simple enough to allow tractable and robust inference. In this paper we will discuss a number of models, of varying sophistication, that fall into a general framework. We will focus on the assumptions and inferential components that are common to these models and consider the specific models only briefly. In particular, we will state the inference algorithms for each model without derivation or proof; the derivations, as well as measures of performance, will appear elsewhere.

## 2 DATA COLLECTION

The algorithms that appear in this paper are likely to be of general applicability. They have been developed, however, with reference to data collected from the parietal cortex of adult rhesus macaques using tetrodes (Pezaris *et al* 1997).

The tetrode is a bundle of four individually insulated $13\mu$m-diameter wires twisted together and cut so that the exposed ends lie close together. The potential on each wire is amplified (custom electronics), low-pass filtered (9-pole Bessel filter, $f_c = 6.4$ kHz) to prevent aliasing, and digitized ($f_s$ between 12.8 and 20 kHz) (filters and A/D converter from Tucker Davis Technologies). This data stream is recorded to digital media; subsequent operations are currently performed off-line.

In preparation for inference, candidate events (where at least one cell fired) are identified in the data stream. The signal is digitally high-pass filtered ($f_c = 0.05f_s$) and the root-mean-square (RMS) amplitude on each channel is calculated. This value is an upper bound on the noise power, and approaches the actual value when the firing rates of resolvable cells are low. Epochs where the signal rises above three times the RMS amplitude for two consecutive signals are taken to be spike events. The signal is upsampled in the region of each such threshold crossing, and the time of the maximal subsequent peak across all channels is determined to within one-tenth of a sample. A short section is then extracted at the original $f_s$ such that this *peak time* falls at a fixed position in the extracted segment. One such waveform is extracted for each threshold crossing.

## 3 GENERATIVE FRAMEWORK

Our basic model is as follows. The recorded potential trace $V(t)$ is the sum of influences that are due to resolvable *foreground* cells (which have a relatively large effect) and a *background* noise process. We write

$$V(t) = \sum_{\tau} \left( c_1^\tau S_1^\tau(t - \tau) + c_2^\tau S_2^\tau(t - \tau) + \cdots \right) + \eta(t) \tag{1}$$

Here, $c_m^\tau$ is an indicator variable that takes the value 1 if the $m$th cell fires at time $\tau$ and 0 otherwise. If cell $m$ fires at $\tau$ it adds a deflection of shape $S_m^\tau(t - \tau)$ to the recorded potential. The effect of all background neural sources, and any electrical noise, is gathered into a single term $\eta(t)$. For a multichannel probe, such as a tetrode, all of $V(t)$, $\eta(t)$ and $S_m^\tau(t)$ are vector-valued. Note that we have indexed

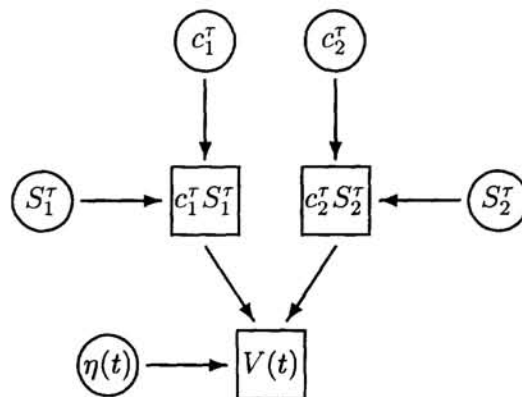

Figure 1: Schematic graph of the general framework.

the spike shapes from the $m$th cell by time; this allows us to model changes in the spike waveform due to intrinsic biophysical processes (such as sodium inactivation during a burst of spikes) as separate to the additive background process. We will discuss models where the choice of $S_m^\tau$ is purely stochastic, as well as models in which both the probability of firing and the shape of the action potential depend on the recent history of the cell.

It will be useful to rewrite (1) in terms of the event waveforms described in section 2. At times $\tau$ when no foreground cell fires all the $c_m^\tau$ are zero. We index the remaining times (when at least one cell fired) by $i$ and write $c_m^i$ for $c_m^\tau$ at $\tau^i$ (similarly for $S_m^i$) to obtain

$$V(t) = \sum_i \left( c_1^i S_1^i(t - \tau^i) + c_2^i S_2^i(t - \tau^i) + \cdots \right) + \eta(t) \qquad (2)$$

This basic model is sketched, for the case of two cells, in figure 1. Circles represent stochastic variables and squares deterministic functions, while arrows indicate conditional or functional dependence. We have not drawn nodes for $\theta_\eta$ and $\theta$. The representation chosen is similar to, and motivated by, a directed acyclic graph (DAG) model of the generative distribution. For clarity, we have not drawn edges that represent dependencies across time steps; the measurement $V(t)$ depends on many nearby values of $S_m^\tau$ and $c_m^\tau$, and $\eta(t)$ may be autocorrelated. We will continue to omit these edges, even when we later show connections in time between $c_m^\tau$ and $S_m^\tau$.

## 4   INFERENCE

We have two statistical objectives. The first is model selection, which includes the choice of the number of cells in the foreground. The second is inference: finding good estimates for the $c_m^\tau$ given the measured $V(t)$. We will have little to say on the subject of model selection in this paper, besides making the observation that standard techniques such as cross-validation, penalized likelihood or approximation of the marginal likelihood (or "evidence") are all plausible approaches. We will instead focus on the inference of the spike times.

Rather than calculating the marginalized posterior for the $c_m^\tau$ we will find the distribution conditioned on the most probable values of the other variables. This is a common approximation to the true posterior (compare Lewicki (1994)).

A simple property of the data allows us to estimate the most probable values of

the parameters in stages; times at which at least one foreground cell fires can be identified by a threshold, as described in section 2. We can then estimate the noise parameters $\theta_\eta$ by looking at segments of the signal with no foreground spikes, the waveform distribution and firing time parameters $\theta$ from the collection of spike events, and finally the spike times $c_m^\tau$ and the waveforms $S_m^\tau$ by a filtering process applied to the complete data $V(t)$ given these model parameters.

## 4.1 NOISE

We study the noise distribution as follows. We extract 1ms segments from a band-passed recording sampled at 16 kHz from a four-channel electrode, avoiding the foreground spikes identified as in section 2. Each segment is thus a 64-dimensional object. We find the principal components of the ensemble of such vectors, and construct histograms of the projections of the vectors in these directions. A few of these histograms are shown on a log-scale in figure 2 (points), as well as a zero-mean Gaussian fit to the distribution projected along the same axes (lines). It is clear that the Gaussian is a reasonable description, although a slight excess in kurtosis is visible in the higher principal components.

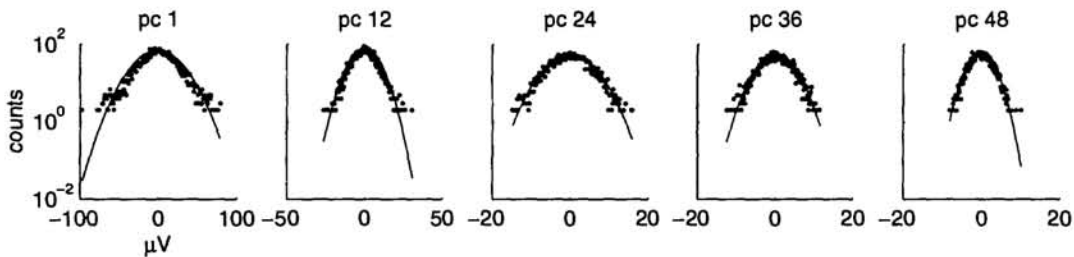

Figure 2: Distribution of background noise.

The noise parameters are now seen to be the covariance of the noise, $\Sigma_\eta$ (we represent it as a covariance *matrix* taken over the length of a spike). In general, we can fit an autoregressive process description to the background and apply a filter that will whiten the noise. This will prove to be quite useful during the filtering stages.

## 4.2 WAVEFORM PARAMETERS

We can make some general remarks about the process of inferring the parameters of the models for $S_m^\tau$ and $c_m^\tau$. Specific models and their inference algorithms will appear in section 5.

The models will, in general, be fit to the collection of segments extracted and aligned as described in section 2. At other times they have no influence on the waveform recorded. We will represent these segments by $V^i$, implying a connection to the firing events $\tau^i$ used in (2). It should be borne in mind that the threshold-based trigger scheme will not exactly identify all of the true $\tau^i$ correctly.

We will assume that each segment represents a single $S_m$, that is, that no two cells fire at times close enough for their spike waveforms to overlap. This is an unreasonable assumption; we can shore it up partially by eliminating from our collection of $V^i$ segments that appear heuristically to contain overlaps (for example, double-peaked waveforms). Ultimately, however, we will need to make our inference procedure robust enough that the parameters describing the model are well estimated despite the errors in the data.

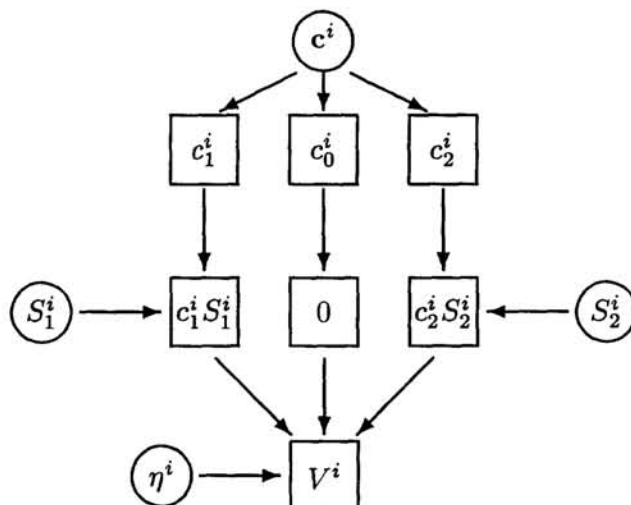

Figure 3: The mixture model for $V^i$.

The advantage to making this assumption is that the overall model for the distribution of the $V^i$ becomes a mixture: a single control variable $\mathbf{c}^i$ sets exactly one of the $c_m^i$ to 1. $V^i$ is then drawn from the distribution of waveforms for the selected cell, convolved with the noise. This is a formal statement of the "clustering" approach to spike-sorting. Mixture models such as these are easy to fit using the Expectation-Maximization (EM) algorithm (Dempster *et al* 1977). We will also consider models with additional latent state variables, which are used to describe the distributions of the $S_m$ and $c_m$, where again EM will be of considerable utility.

The measured ensemble $V^i$ will be incorrect on a number of counts. The threshold may make either false positive or false negative errors in selecting $\tau^i$, and some of the identified $V^i$ will represent overlaps. We can use heuristics to minimize such errors, but need to account for any remaining outliers in our models. We do so by introducing additional mixture components. Segments of noise that are incorrectly identified as foreground events are handled by an explicit zero mixture component whose variability is entirely due to the background noise. Overlaps are handled by providing very broad low-probability components spanning large areas in waveform space; clusters of overlap waveforms are likely to be diffuse and sparse.

The mixture model is sketched in figure 3. In the basic model the variables are chosen independently for each cross-threshold event. The dynamic models discussed below will introduce dependencies in time.

## 4.3   SPIKE TIMES

In our final stage of inference, we make estimates of the $c_m^\tau$ given the $V(t)$ and the most-probable parameters fit in the previous two stages. This is exactly the signal detection problem of identifying pulses (perhaps with random or else adapting parameters) in Gaussian noise of known covariance. Solutions to this are well known (McDonough and Whalen 1995) and easily adapted to the problem at hand (Sahani *et al* 1998).

# 5  SPECIFIC MODELS

Finally, we describe examples of models that may be used within this framework. As stated before, in this brief catalog we summarize the motivation for each, and state without derivation or proof the algorithms for inference. The details of these algorithms, as well as tests of performance, will appear elsewhere.

## 5.1  CONSTANT WAVEFORM

The simplest model is one in which we take the waveform of the $m$th cell to remain unchanged and the firing probability of each cell to be constant. In this case we drop the index $\tau$ or $i$ on the waveform shape and just write $S_m(t - \tau^i)$. We write $p_m$ for the probability that a given event is due to the $m$th cell firing. The mixture model is then a mixture of multivariate Gaussian distributions, each with covariance $\Sigma_\eta$, mean $S_m$ and mixture fraction $p_m$. The EM algorithm for such a mixture is well known (Nowlan 1990).

Given parameters $\theta^{(n)} = \{S_m^{(n)}, p_m^{(n)}\}$ from the $n$th iteration, we find the expected values of the $c_m^i$ (called the *responsibilities*),

$$r_m^i = \mathcal{E}[c_m^i \mid \{V^i\}, \theta^{(n)}] = \frac{p_m^{(n)} N(V^i; S_m^{(n)}, \Sigma_\eta)}{\sum_{\tilde{m}} p_{\tilde{m}}^{(n)} N(V^i; S_{\tilde{m}}^{(n)}, \Sigma_\eta)}, \tag{3}$$

and then reestimate the parameters from the data weighted by the responsibilities.

$$p_m^{(n+1)} = \frac{\sum_i r_m^i}{N} \; ; \; S_m^{(n+1)} = \frac{\sum_i r_m^i V^i}{\sum_i r_m^i}. \tag{4}$$

## 5.2  REFRACTORY FIRING

A simple modification to this scheme can be used to account for the refractory period between spikes from the same cell (Sahani *et al* 1998). The model is similar to the Gaussian mixture above, except that the choice of mixture component is no longer independent for each waveform. If the waveforms arrive within a refractory period they cannot have come from the same cell. This leads to the altered responsibilities:

$$s_m^i = \frac{r_m^i}{Z^i} \prod_{j:(i,j) \text{ refractory}} (1 - r_m^j) \tag{5}$$

where $Z$ is a normalizing constant.

The M step here is identical to (4), with the responsibilities $s_m^i$ replacing the $r_m^i$.

## 5.3  STATIC MIXTURE

As we have suggested above, the waveform of the $m$th cell is not, in fact, unchanged each time the cell fires. Variability in excess of the additive background noise is introduced by changes in the biophysical properties of the cell (due to recent firing patterns, or external modulators) as well as by background activity that may be correlated with foreground events. We can attempt to model this variability as giving rise to a discrete set of distinct waveforms, which are then convolved with the previously measured noise covariance to obtain the distribution of measurements. In effect, we are tiling an irregularly shaped distribution with a mixture of Gaussians

of fixed shape, $\Sigma_\eta$. We obtain a hierarchical mixture distribution in which each component corresponding to a cell is itself is a mixture of Gaussians. Given a particular hierarchical arrangement the parameters can be fit exactly as above.

While this approach seems attractive, it suffers from the flaw that model selection is not well defined. In particular, the hierarchical mixture is equivalent in terms of likelihood and parameters to a single-layer, flat, mixture. To avoid this problem we may introduce a prior requiring that the Gaussian components from a single cell overlap, or otherwise lie close together. It is, however, difficult to avoid excessive sensitivity to such a prior.

## 5.4  DYNAMICAL MIXTURE

An alternative approach is to replace the independent transitions between the components of the mixture distribution of a single cell with a dynamical process that reflects the manner in which both firing probability and waveform shape depend on the recent history of the cell. In this view we may construct a mixture of hidden Markov models (HMMs), one for each cell. Our earlier mixture assumption now means that the models must be coupled so that on any one time step at most one makes a transition to a state corresponding to firing. This structure may be thought of as a special case of the factorial HMM discussed by Gharamani and Jordan (1997).

The general model is known to be intractable. In this special case, however, the standard forward-backward procedure for a single HMM can be modified to operate on reponsibility-weighted data, where the reponsibilities are themselves calculated during the forward phase. This is empirically found to provide an effective E step. The M step is then straightforward.

## Acknowledgements

This work has benefited considerably from important discussions with both Bill Bialek and Sam Roweis. John Hopfield has provided invaluable advice and mentoring to MS. We thank Jennifer Linden and Philip Sabes for useful comments on an earlier version of the manuscript. Funding for various components of the work has been provided by the Keck Foundation, the Sloan Center for Theoretical Neuroscience at Caltech, the Center for Neuromorphic Systems Engineering at Caltech, and the National Institutes of Health.

## References

Dempster, A. P., N. M. Laird, and D. B. Rubin (1977). *J. Royal Stat. Soc. B 39*, 1–38.

Fee, M. S., P. P. Mitra, and D. Kleinfeld (1996). *J. Neurophys. 76*(3), 3823–3833.

Gharamani, Z. and M. I. Jordan (1997). *Machine Learning 29*, 245–275.

Lewicki, M. S. (1994). *Neural Comp. 6*(5), 1005–1030.

McDonough, R. N. and A. D. Whalen (1995). *Detection of Signals in Noise* (2nd ed.). San Diego: Academic Press.

Nowlan, S. J. (1990). In D. S. Touretzky (Ed.), *Advances in Neural Information Processing Systems 2*, San Mateo, CA. Morgan Kaufmann.

Pezaris, J. S., M. Sahani, and R. A. Andersen (1997). In J. M. Bower (Ed.), *Computational Neuroscience: Trends in Research, 1997.*

Sahani, M., J. S. Pezaris, and R. A. Andersen (1998). In J. M. Bower (Ed.), *Computational Neuroscience: Trends in Research, 1998.*